# NEURAL NETWORK VISUALIZATION

Jakub Wejchert
Gerald Tesauro
IBM Research
T.J. Watson Research
Center
Yorktown Heights
NY 10598

## ABSTRACT

We have developed graphics to visualize static and dynamic information in layered neural network learning systems. Emphasis was placed on creating new visuals that make use of spatial arrangements, size information, animation and color. We applied these tools to the study of back-propagation learning of simple Boolean predicates, and have obtained new insights into the dynamics of the learning process.

## 1 INTRODUCTION

Although neural network learning systems are being widely investigated by many researchers via computer simulations, the graphical display of information in these simulations has received relatively little attention. In other fields such as fluid dynamics and chaos theory, the development of "scientific visualization" techniques (1,3) have proven to be a tremendously useful aid to research, development, and education. Similar benefits should result from the application of these techniques to neural networks research.

In this article, several visualization methods are introduced to investigate learning in neural networks which use the back-propagation algorithm. A multi-window

environment is used that allows different aspects of the simulation to be displayed simultaneously in each window.

As an application, the toolkit is used to study small networks learning Boolean functions. The animations are used to observe the emerging structure of connection strengths, to study the temporal behaviour, and to understand the relationships and effects of parameters. The simulations and graphics can run at real-time speeds.

## 2    VISUAL REPRESENTATIONS

First, we introduce our techniques for representing both the instantaneous dynamics of the learning process, and the full temporal trajectory of the network during the course of one or more learning runs.

### 2.1    The Bond Diagram

In the first of these diagrams, the geometrical structure of a connected network is used as a basis for the representation. As it is of interest to try to see how the internal configuration of weights relates to the problem the network is learning, it is clearly worthwhile to have a graphical representation that explicitly includes weight information integrated with network topology. This differs from "Hinton diagrams" (2), in which data may only be indirectly related to the network structure. In our representation nodes are represented by circles, the area of which are proportional to the threshold values. Triangles or lines are used to represent the weights or their rate of change. The triangles or line segments emanate from the nodes and point toward the connecting nodes. Their lengths indicate the magnitude of the weight or weight derivative. We call this the "bond diagram".

In this diagram, one can look at any node and clearly see the magnitude of the weights feeding into and out of it. Also, a sense of direction is built into the picture since the bonds point to the node that they are connected to. Further, the collection of weights form distinct patterns that can be easily perceived, so that one can also infer global information from the overall patterns formed.

### 2.2    The Trajectory Diagram

A further limitation of Hinton diagrams is that they provide a relatively poor representation of dynamic information. Therefore, to understand more about the dynamics of learning we introduce another visual tool that gives a two-dimensional projection of the weight space of the network. This represents the learning process as a trajectory in a reduced dimensional space. By representing the value of the error function as the color of the point in weight space, one obtains a sense of the contours of the error hypersurface, and the dynamics of the gradient-descent evolution on this hypersurface. We call this the "trajectory diagram".

The scheme is based on the premise that the human user has a good visual notion of vector addition. To represent an n-dimensional point, its axial components are defined as vectors and then are plotted radially in the plane; the vector sum of these is then calculated to yield the point representing the n-dimensional position.

It is obvious that for n > 2 the resultant point is not unique, however, the method does allow one to infer information about families of similar trajectories, make comparisons between trajectories and notice important deviations in behaviour.

## 2.3 Implementation

The graphics software was written in C using X-Windows v. 11. The C code was interfaced to a FORTRAN neural network simulator. The whole package ran under UNIX, on an RT workstation. Using the portability of X-Windows the graphics could be run remotely on different machines using a local area network. Exceecution time was slow for real-time interaction except for very small networks (typically up 30 weights). For larger networks, the Stellar graphics workstation was used, whereby the simulator code could be vectorized and parallelized.

# 3    APPLICATION EXAMPLES

With the graphics we investigated networks learning Boolean functions: binary input vectors were presented to the network through the input nodes, and the teacher signal was set to either 1 or 0. Here, we show networks learning majority, and symmetry functions. The output of the majority function is 1 only if more than half of the input nodes are on; simple symmetry distiguishes between input vectors that are symmetric or anti-symmetric about a central axis; general symmetry identifies perfectly symmetric patterns out of all other permutations. Using the graphics, one can watch how solutions to a particular problem are obtained, how different parameters affect these solutions, and observe stages at which learning decisions are made.

At the start of the simulations the weights are set to small random values. During learning, many example patterns of vectors are presented to the input of the network and weights are adjusted accordingly. Initially the rate of change of weights is small, later as the simulation gets under way the weights change rapidly, until small changes are made as the system moves toward the final solution. Distinct patterns of triangles show the configuration of weights in their final form.

## 3.1 The Majority Function

Figure 1 shows a bond diagram for a network that has learnt the majority function. During the run, many input patterns were presented to the network during which time the weights were changed. The weights evolve from small random values through to an almost uniform set corresponding to the solution of the problem. Towards the end, a large output node is displayed and the magnitudes of all the weights are roughly uniform, indicating that a large bias (or threshold) is required to offset the sum of the weights. Majority is quite a simple problem for the network to learn; more complicated functions require hidden units.

## 3.2 The Simple Symmetry Function

In this case only symmetric or perfectly anti-symmetric patterns are presented and the network is taught to distinguish between these. In solving this problem, the

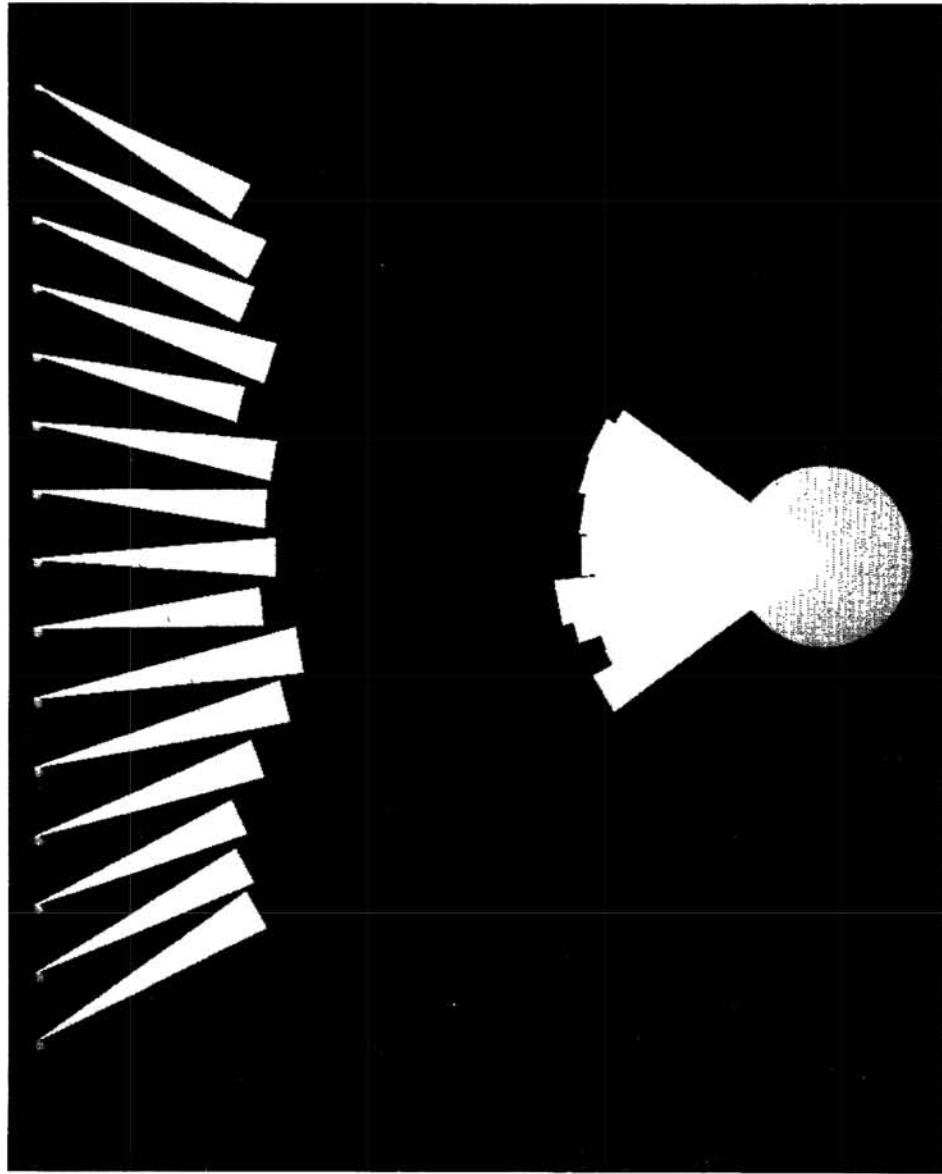

**Figure 1:** A near-final configuration of weights for the majority function. All the weights are positive. The disc corresponds to the threshold of the output unit.

network chose (correctly) that it needs only two units to make the decision whether the input is totally symmetric or totally anti-symmetric. (In fact, any symmetrically separated input pair will work.) It was found that the simple pattern created by the bond representation carries over into the more general symmetry function, where the network must identify perfectly symmetric inputs from all the other permutations.

## 3.3    The General Symmetry Function

Here, the network is required to detect symmtery out of all the possible input patterns. As can be seen from the bond diagram (figure 2) the network has chosen a hierarchical structure of weights to solve the problem, using the basic pattern of weights of simple symmtery. The major decision is made on the outer pair and additional decisions are made on the remaining pairs with decreasing strength. As before, the choice of pairs in the hierarchy depends on the initial random weights. By watching the animations, we could make some observations about the stages of learning. We found that the early behavior was the most critical as it was at this stage that the signs of the weights feeding to the hidden units were determined. At the later stages the relative magnitudes of the weights were adapted.

## 3.4    The Visualization Environment

Figure 3 shows the visualization environment with most of the windows active. The upper window shows the total error, and the lower window the state of the output unit. Typically, the error initially stays high then decreases rapidly and then levels off to zero as final adjustments are made to the weights. Spikes in this curve are due to the method of presenting patterns at random. The state of the output unit initially oscillates and then bifurcates into the two requires output states.

The two extra windows on the right show the trajectory diagrams for the two hidden units. These diagrams are generalizations of phase diagrams: components of a point in a high dimensional space are plotted radially in the plane and treated as vectors whose sum yields a point in the two-dimensional representation. We have found these diagrams useful in observing the trajectories of the two hidden units, in which case they are representations of paths in a six-dimensional weight space. In cases where the network does converge to a correct solution, the paths of the two hidden units either try to match each other (in which case the configurations of the units were identical) or move in opposite directions (in which case the units were opposites).

By contrast, for learning runs which do not converge to global optima we found that usually one of the hidden units followed a normal trajectory whereas the other unit was not able to achieve the appropriate match or anti-match. This is because the signs of the weights to the second hidden unit were not correct and the learning algorithm could not make the necessary adjustments. At a certain point early in learning the unit would travel off on a completely different trajectory. These obser-vations suggest a heuristic that could improve learning by setting initial trajectories in the "correct" directions.

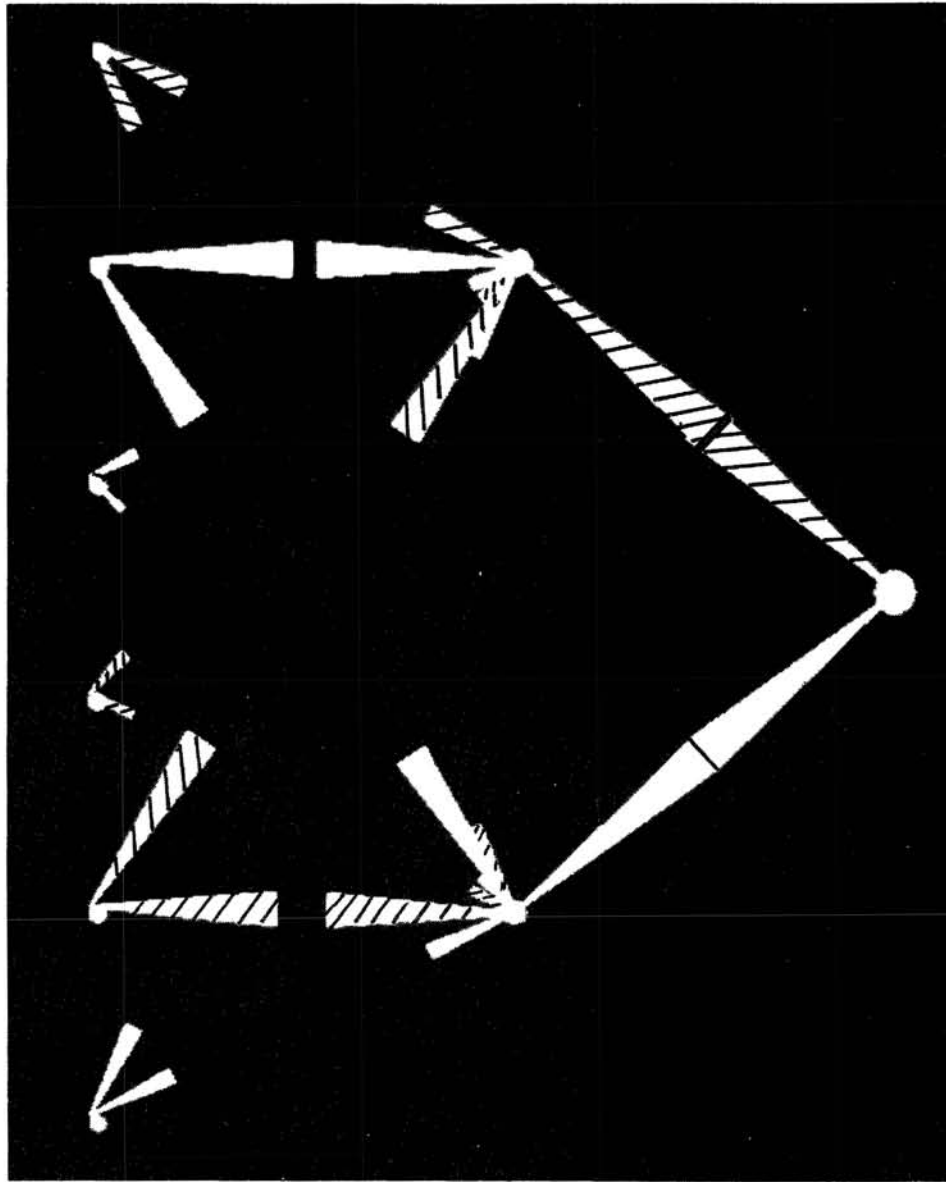

**Figure 2:** The bond diagram for a network that has learnt the symmetry function. There are six input units, two hidden and one output. Weights are shown by bonds emantating from nodes. In the graphics positive and negative weights are colored red and blue respectively. In this grey-scale photo the negative weights are marked with diagonal lines to distiguish them from positive weights.

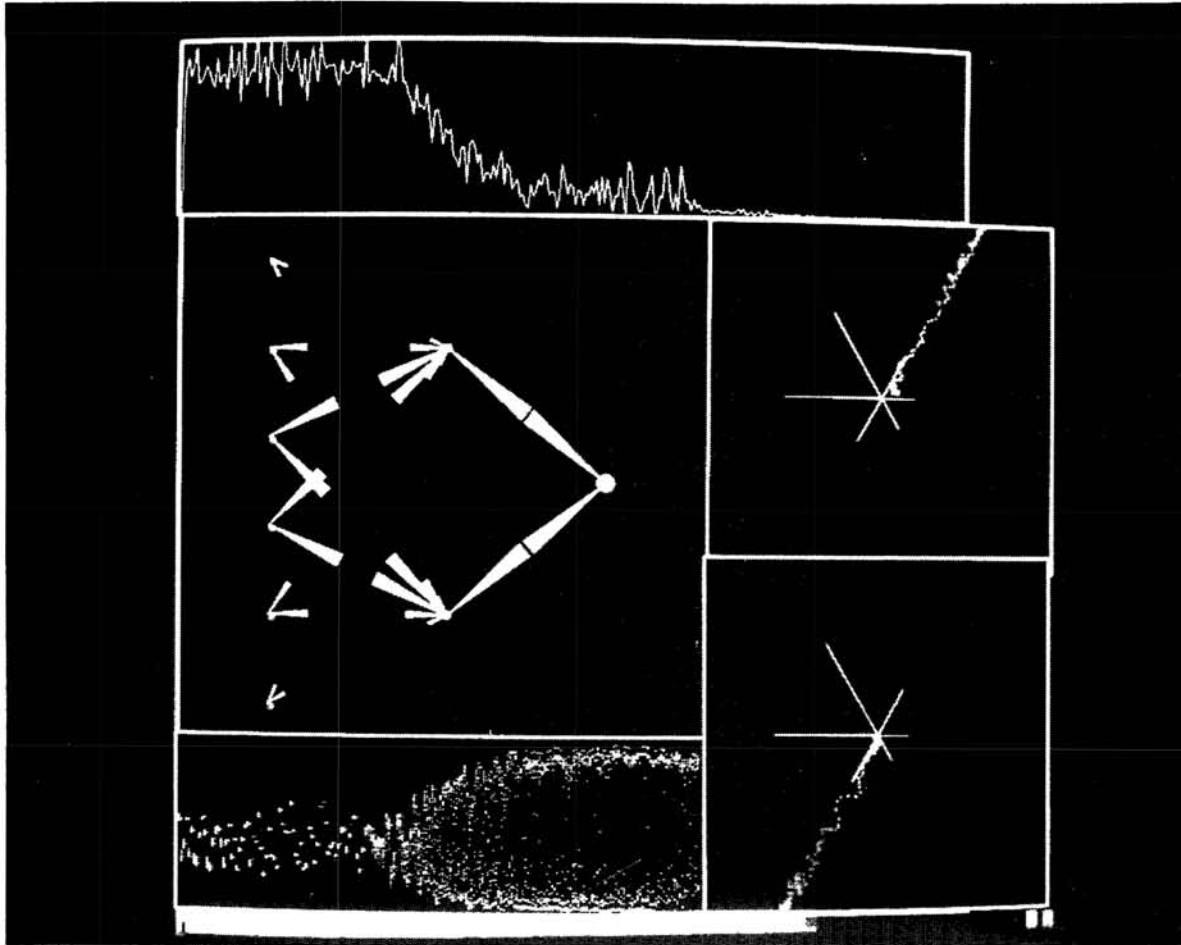

**Figure 3:** An example of the graphics with most of the windows active; the command line appears on the bottom. The central window shows the bond diagram of the General Symmetry function. The upper left window shows the total error, and the lower left window the state of the output unit. The two windows on the right show the trajectory diagrams for the two hidden units. The "spokes" in this diagram correspond to the magnitude of the weights. The trace of dots are the paths of the two units in weight space.

In general, the trajectory diagram has similar uses to a conventional phase plot: it can distinguish between different regions of configuration space; it can be used to detect critical stages of the dynamics of a system; and it gives a "trace" of its time evolution.

# 4  CONCLUSION

A set of computer graphics visualization programs have been designed and interfaced to a back-propagation simulator. Some new visualization tools were introduced such as the bond and trajectory diagrams. These and other visual tools were integrated into an interactive multi-window environment.

During the course of the work it was found that the graphics was useful in a number of ways: in giving a clearer picture of the internal representation of weights, the effects of parameters, the detection of errors in the code, and pointing out aspects of the simulation that had not been expected beforehand. Also, insight was gained into principles of designing graphics for scientific processes.

It would be of interest to extend our visualization techniques to include large networks with thousands of nodes and tens of thousands of weights. We are currently examining a number of alternative techniques which are more appropriate for large data-set regimes.

**Acknowledgements**

We wish to thank Scott Kirkpatrick for help and encouragment during the project. We also thank members of the visualization lab and the animation lab for use of their resources.

**References**

(1) McCormick B H, DeFanti T A Brown M D (Eds), "Visualization in Scientific Computing" Computer Graphics 21, 6, November (1987). See also "Visualization in Scientific Computing-A Synopsis", IEEE Computer Graphics and Applications, July (1987).

(2) Rumelhart D E, McClelland J L, "Parallel Distributed Processing: Explorations in the Microstructure of Cognition. Volume 1" MIT Press, Cambridge, MA (1986).

(3) Tufte E R, "The Visual Display of Quantitative Information", Graphic Press, Chesire, CT (1983).
